# Divergences, surrogate loss functions and experimental design

**XuanLong Nguyen**
University of California
Berkeley, CA 94720
xuanlong@cs.berkeley.edu

**Martin J. Wainwright**
University of California
Berkeley, CA 94720
wainwrig@eecs.berkeley.edu

**Michael I. Jordan**
University of California
Berkeley, CA 94720
jordan@cs.berkeley.edu

## Abstract

In this paper, we provide a general theorem that establishes a correspondence between surrogate loss functions in classification and the family of $f$-divergences. Moreover, we provide constructive procedures for determining the $f$-divergence induced by a given surrogate loss, and conversely for finding all surrogate loss functions that realize a given $f$-divergence. Next we introduce the notion of universal equivalence among loss functions and corresponding $f$-divergences, and provide necessary and sufficient conditions for universal equivalence to hold. These ideas have applications to classification problems that also involve a component of experiment design; in particular, we leverage our results to prove consistency of a procedure for learning a classifier under decentralization requirements.

## 1 Introduction

A unifying theme in the recent literature on classification is the notion of a *surrogate loss function*—a convex upper bound on the 0-1 loss. Many practical classification algorithms can be formulated in terms of the minimization of surrogate loss functions; well-known examples include the support vector machine (hinge loss) and Adaboost (exponential loss). Significant progress has been made on the theoretical front by analyzing the general statistical consequences of using surrogate loss functions [e.g., 2, 10, 13].

These recent developments have an interesting historical antecedent. Working in the context of experimental design, researchers in the 1960's recast the (intractable) problem of minimizing the probability of classification error in terms of the maximization of various surrogate functions [e.g., 5, 8]. Examples of experimental design include the choice of a quantizer as a preprocessor for a classifier [12], or the choice of a "signal set" for a radar system [5]. The surrogate functions that were used included the Hellinger distance and various forms of KL divergence; maximization of these functions was proposed as a criterion for the choice of a design. Theoretical support for this approach was provided by a classical theorem on the comparison of experiments due to Blackwell [3]. An important outcome of this line of work was the definition of a general family of "$f$-divergences" (also known as "Ali-Silvey distances"), which includes Hellinger distance and KL divergence as special cases [1, 4].

In broad terms, the goal of the current paper is to bring together these two literatures, in particular by establishing a correspondence between the family of surrogate loss functions and the family of $f$-divergences. Several specific goals motivate us in this regard: (1) different $f$-divergences are related by various well-known inequalities [11], so that a correspondence between loss functions and $f$-divergences would allow these inequalities to be harnessed in analyzing surrogate loss functions; (2) a correspondence could allow the definition of interesting equivalence classes of losses or divergences; and (3) the problem of experimental design, which motivated the classical research on $f$-divergences, provides new venues for applying the loss function framework from machine learning. In particular, one natural extension—and one which we explore towards the end of this paper—is in requiring consistency not only in the choice of an optimal discriminant function but also in the choice of an optimal experiment design.

The main technical contribution of this paper is to state and prove a general theorem relating surrogate loss functions and $f$-divergences. [1] We show that the correspondence is quite strong: any surrogate loss induces a corresponding $f$-divergence, and any $f$-divergence satisfying certain conditions corresponds to a family of surrogate loss functions. Moreover, exploiting tools from convex analysis, we provide a constructive procedure for finding loss functions from $f$-divergences. We also introduce and analyze a notion of *universal equivalence* among loss functions (and corresponding $f$-divergences). Finally, we present an application of these ideas to the problem of proving consistency of classification algorithms with an additional decentralization requirement.

## 2 Background and elementary results

Consider a covariate $X \in \mathcal{X}$, where $\mathcal{X}$ is a compact topological space, and a random variable $Y \in \mathcal{Y} := \{-1, +1\}$. The space $(X \times Y)$ is assumed to be endowed with a Borel regular probability measure $P$. In this paper, we consider a variant of the standard classification problem, in which the decision-maker, rather than having direct access to $X$, only observes some variable $Z \in \mathcal{Z}$ that is obtained via conditional probability $Q(Z|X)$. The stochastic map $Q$ is referred to as an *experiment* in statistics; in the signal processing literature, where $\mathcal{Z}$ is generally taken to be discrete, it is referred to as a *quantizer*. We let $\mathcal{Q}$ denote the space of all stochastic $Q$ and let $\mathcal{Q}_0$ denote its deterministic subset.

Given a fixed experiment $Q$, we can formulate a standard binary classification problem as one of finding a measurable function $\gamma \in \Gamma := \{\mathcal{Z} \to \mathbb{R}\}$ that minimizes the *Bayes risk* $P(Y \neq \mathrm{sign}(\gamma(Z)))$. Our focus is the broader question of determining both the classifier $\gamma \in \Gamma$, as well as the experiment choice $Q \in \mathcal{Q}$ so as to minimize the Bayes risk.

The Bayes risk corresponds to the expectation of the 0-1 loss. Given the non-convexity of this loss function, it is natural to consider a surrogate loss function $\phi$ that we optimize in place of the 0-1 loss. We refer to the quantity $R_\phi(\gamma, Q) := \mathbb{E}\phi(Y\gamma(Z))$ as the *$\phi$-risk*. For each fixed quantization rule $Q$, the optimal $\phi$ risk (as a function of $Q$) is defined as follows:

$$R_\phi(Q) := \inf_{\gamma \in \Gamma} R_\phi(\gamma, Q). \tag{1}$$

Given priors $q = P(Y = -1)$ and $p = P(Y = 1)$, define nonnegative measures $\mu$ and $\pi$:

$$\mu(z) \;=\; P(Y = 1, Z = z) = p \int_x Q(z|x) dP(x|Y = 1)$$

$$\pi(z) \;=\; P(Y = -1, Z = z) = q \int_x Q(z|x) dP(x|Y = -1).$$

As a consequence of Lyapunov's theorem, the space of $\{(\mu, \pi)\}$ obtained by varying $Q \in \mathcal{Q}$ (or $\mathcal{Q}_0$) is both compact and convex (see [12] for details). For simplicity, we assume that the space $\mathcal{Q}$ of $Q$ is restricted such that both $\mu$ and $\pi$ are strictly positive measures.

One approach to choosing $Q$ is to define an $f$-divergence between $\mu$ and $\pi$; indeed this is the classical approach referred to earlier [e.g., 8]. Rather than following this route, however, we take an alternative path, setting up the problem in terms of $\phi$-risk and optimizing out the discriminant function $\gamma$. Note in particular that the $\phi$-risk can be represented in terms of the measures $\mu$ and $\pi$ as follows:

$$R_\phi(\gamma, Q) \;=\; \sum_z \phi(\gamma(z))\mu(z) + \phi(-\gamma(z))\pi(z). \qquad (2)$$

This representation allows us to compute the optimal value for $\gamma(z)$ for all $z \in \mathcal{Z}$, as well as the optimal $\phi$ risk for a fixed $Q$. We illustrate this calculation with several examples:

**0-1 loss.** If $\phi$ is 0-1 loss, then $\gamma(z) = \text{sign}(\mu(z) - \pi(z))$. Thus the optimal Bayes risk given a fixed $Q$ takes the form: $R_{bayes}(Q) = \sum_{z \in \mathcal{Z}} \min\{\mu(z), \pi(z)\} = \frac{1}{2} - \frac{1}{2}\sum_{z \in \mathcal{Z}} |\mu(z) - \pi(z)| =: \frac{1}{2}(1 - V(\mu, \pi))$, where $V(\mu, \pi)$ denotes the variational distance between two measures $\mu$ and $\pi$.

**Hinge loss.** Let $\phi_{hinge}(y\gamma(z)) = (1 - y\gamma(z))_+$. In this case $\gamma(z) = \text{sign}(\mu(z) - \pi(z))$ and the optimal risk takes the form: $R_{hinge}(Q) = \sum_{z \in \mathcal{Z}} 2\min\{\mu(z), \pi(z)\} = 1 - \sum_{z \in \mathcal{Z}} |\mu(z) - \pi(z)| = 1 - V(\mu, \pi) = 2R_{bayes}(Q)$.

**Least squares loss.** Letting $\phi_{sqr}(y\gamma(z)) = (1 - y\gamma(z))^2$, we have $\gamma(z) = \frac{\mu(z) - \pi(z)}{\mu(z) + \pi(z)}$. The optimal risk takes the form: $R_{sqr}(Q) = \sum_{z \in \mathcal{Z}} \frac{4\mu(z)\pi(z)}{\mu(z) + \pi(z)} = 1 - \sum_{z \in \mathcal{Z}} \frac{(\mu(z) - \pi(z))^2}{\mu(z) + \pi(z)} =: 1 - \Delta(\mu, \pi)$, where $\Delta(\mu, \pi)$ denotes the *triangular discrimination* distance.

**Logistic loss.** Letting $\phi_{log}(y\gamma(z)) := \log\left(1 + \exp^{-y\gamma(z)}\right)$, we have $\gamma(z) = \log\frac{\mu(z)}{\pi(z)}$. The optimal risk for logistic loss takes the form: $R_{log}(Q) = \sum_{z \in \mathcal{Z}} \mu(z)\log\frac{\mu(z) + \pi(z)}{\mu(z)} + \pi(z)\log\frac{\mu(z) + \pi(z)}{\pi(z)} = \log 2 - KL(\mu||\frac{\mu + \pi}{2}) - KL(\pi||\frac{\mu + \pi}{2}) =: \log 2 - C(\mu, \pi)$, where $C(U, V)$ denotes the *capacitory discrimination* distance.

**Exponential loss.** Letting $\phi_{exp}(y\gamma(z)) = \exp(-y\gamma(z))$, we have $\gamma(z) = \frac{1}{2}\log\frac{\mu(z)}{\pi(z)}$. The optimal risk for exponential loss takes the form: $R_{exp}(Q) = \sum_{z \in \mathcal{Z}} 2\sqrt{\mu(z)\pi(z)} = 1 - \sum_{z \in \mathcal{Z}} (\sqrt{\mu(z)} - \sqrt{\pi(z)})^2 = 1 - 2h^2(\mu, \pi)$, where $h(\mu, \pi)$ denotes the Hellinger distance between measures $\mu$ and $\pi$.

All of the distances given above (e.g., variational, Hellinger) are all particular instances of $f$-divergences. This fact points to an interesting correspondence between optimized $\phi$-risks and $f$-divergences. How general is this correspondence?

## 3  The correspondence between loss functions and $f$-divergences

In order to resolve this question, we begin with precise definitions of $f$-divergences, and surrogate loss functions. A *f-divergence functional* is defined as follows [1, 4]:

**Definition 1.** *Given any continuous convex function $f : [0, +\infty) \to \mathbb{R} \cup \{+\infty\}$, the f-divergence between measures $\mu$ and $\pi$ is given by $I_f(\mu, \pi) := \sum_z \pi(z) f\left(\frac{\mu(z)}{\pi(z)}\right)$.*

For instance, the variational distance is given by $f(u) = |u - 1|$, KL divergence by $f(u) = u\log u$, triangular discrimination by $f(u) = (u - 1)^2/(u + 1)$, and Hellinger distance by $f(u) = \frac{1}{2}(\sqrt{u} - 1)^2$.

**Surrogate loss $\phi$.** First, we require that any *surrogate* loss function $\phi$ is continuous and convex. Second, the function $\phi$ must be *classification-calibrated* [2], meaning that for any $a, b \geq 0$ and $a \neq b$, $\inf_{\alpha:\alpha(a-b)<0} \phi(\alpha)a + \phi(-\alpha)b > \inf_{\alpha \in \mathbb{R}} \phi(\alpha)a + \phi(-\alpha)b$. It can be shown [2] that in the convex case $\phi$ is classification-calibrated if and only if it is differentiable at 0 and $\phi'(0) < 0$. Lastly, let $\alpha^* = \inf_\alpha \{\phi(\alpha) = \inf \phi\}$. If $\alpha^* < +\infty$, then for any $\delta > 0$, we require that $\phi(\alpha^* - \delta) \geq \phi(\alpha^* + \delta)$. The interpretation of the last assumption is that one should penalize deviations away from $\alpha^*$ in the negative direction at least as strongly as deviations in the positive direction; this requirement is intuitively reasonable given the margin-based interpretation of $\alpha$.

**From $\phi$-risk to $f$-divergence.** We begin with a simple result that formalizes how any $\phi$-risk induces a corresponding $f$-divergence. More precisely, the following lemma proves that the optimal $\phi$ risk for a fixed $Q$ can be written as the negative of an $f$ divergence.

**Lemma 2.** *For each fixed $Q$, let $\gamma_Q$ denote the optimal decision rule. The $\phi$ risk for $(Q, \gamma_Q)$ is an $f$-divergence between $\mu$ and $\pi$ for some convex function $f$:*

$$R_\phi(Q) = -I_f(\mu, \pi). \tag{3}$$

*Proof.* The optimal $\phi$ risk takes the form:

$$R_\phi(Q) = \sum_{z \in \mathcal{Z}} \inf_\alpha (\phi(\alpha)\mu(z) + \phi(-\alpha)\pi(z)) = \sum_z \pi(z) \inf_\alpha \left( \phi(-\alpha) + \phi(\alpha)\frac{\mu(z)}{\pi(z)} \right).$$

For each $z$ let $u = \frac{\mu(z)}{\pi(z)}$, then $\inf_\alpha(\phi(-\alpha) + \phi(\alpha)u)$ is a concave function of $u$ (since minimization over a set of linear function is a concave function). Thus, the claim follows by defining (for $u \in \mathbb{R}$)

$$f(u) := -\inf_\alpha(\phi(-\alpha) + \phi(\alpha)u). \tag{4}$$

**From $f$-divergence to $\phi$-risk.** In the remainder of this section, we explore the converse of Lemma 2. Given a divergence $I_f(\mu, \pi)$ for some convex function $f$, does there exist a loss function $\phi$ for which $R_\phi(Q) = -I_f(\mu, \pi)$? In the following, we provide a precise characterization of the set of $f$-divergences that can be realized in this way, as well as a constructive procedure for determining all $\phi$ that realize a given $f$-divergence.

Our method requires the introduction of several intermediate functions. First, let us define, for each $\beta$, the inverse mapping $\phi^{-1}(\beta) := \inf\{\alpha : \phi(\alpha) \leq \beta\}$, where $\inf \emptyset := +\infty$. Using the function $\phi^{-1}$, we then define a new function $\Psi : \mathbb{R} \to \overline{\mathbb{R}}$ by

$$\Psi(\beta) \quad := \quad \begin{cases} \phi(-\phi^{-1}(\beta)) & \text{if } \phi^{-1}(\beta) \in \mathbb{R}, \\ +\infty & \text{otherwise.} \end{cases} \tag{5}$$

Note that the domain of $\Psi$ is $\text{Dom}(\Psi) = \{\beta \in \mathbb{R} : \phi^{-1}(\beta) \in \mathbb{R}\}$. Define

$$\beta_1 := \inf\{\beta : \Psi(\beta) < +\infty\} \text{ and } \beta_2 := \inf\{\beta : \Psi(\beta) = \inf \Psi\}. \tag{6}$$

It is simple to check that $\inf \phi = \inf \Psi = \phi(\alpha^*)$, and $\beta_1 = \phi(\alpha^*)$, $\beta_2 = \phi(-\alpha^*)$. Furthermore, $\Psi(\beta_2) = \phi(\alpha^*) = \beta_1$, $\Psi(\beta_1) = \phi(-\alpha^*) = \beta_2$. With this set-up, the following lemma captures several important properties of $\Psi$:

**Lemma 3.**     *(a)* $\Psi$ *is strictly decreasing in* $(\beta_1, \beta_2)$. *If $\phi$ is decreasing, then $\Psi$ is also decreasing in* $(-\infty, +\infty)$. *In addition,* $\Psi(\beta) = +\infty$ *for* $\beta < \beta_1$.

    *(b)* $\Psi$ *is convex in* $(-\infty, \beta_2]$. *If $\phi$ is decreasing, then $\Psi$ is convex in* $(-\infty, +\infty)$.

    *(c)* $\Psi$ *is lower semi-continuous, and continuous in its domain.*

    *(d) There exists* $u^* \in (\beta_1, \beta_2)$ *such that* $\Psi(u^*) = u^*$.

*(e) There holds $\Psi(\Psi(\beta)) = \beta$ for all $\beta \in (\beta_1, \beta_2)$.*

The connection between $\Psi$ and an $f$-divergence arises from the following fact. Given the definition (5) of $\Psi$, it is possible to show that

$$f(u) = \sup_{\beta \in \mathbb{R}}(-\beta u - \Psi(\beta)) = \Psi^*(-u), \qquad (7)$$

where $\Psi^*$ denotes the conjugate dual of the function $\Psi$. Hence, if $\Psi$ is a lower semicontinuous convex function, it is possible to recover $\Psi$ from $f$ by means of convex duality [9]: $\Psi(\beta) = f^*(-\beta)$. Thus, equation (5) provides means for recovering a loss function $\phi$ from $\Psi$. Indeed, the following theorem provides a constructive procedure for finding all such $\phi$ when $\Psi$ satisfies necessary conditions specified in Lemma 3:

**Theorem 4.** *(a) Given a lower semicontinuous convex function $f : \mathbb{R} \to \overline{\mathbb{R}}$, define:*

$$\Psi(\beta) = f^*(-\beta). \qquad (8)$$

*If $\Psi$ is a decreasing function satisfying the properties specified in parts (c), (d) and (e) of Lemma 3, then there exist convex continuous loss function $\phi$ for which* (3) *and* (4) *hold.*

*(b) More precisely, all such functions $\phi$ are of the form: For any $\alpha \geq 0$,*

$$\phi(\alpha) = \Psi(g(\alpha + u^*)), \quad and \quad \phi(-\alpha) = g(\alpha + u^*), \qquad (9)$$

*where $u^*$ satisfies $\Psi(u^*) = u^*$ for some $u^* \in (\beta_1, \beta_2)$ and $g : [u^*, +\infty) \to \overline{\mathbb{R}}$ is any increasing continuous convex function such that $g(u^*) = u^*$. Moreover, $g$ is differentiable at $u^*+$ and $g'(u^*+) > 0$.*

One interesting consequence of Theorem 4 that any realizable $f$-divergence can in fact be obtained from a fairly large set of $\phi$ loss functions. More precisely, examining the statement of Theorem 4(b) reveals that for $\alpha \leq 0$, we are free to choose a function $g$ that must satisfy only mild conditions; given a choice of $g$, then $\phi$ is specified for $\alpha > 0$ accordingly by equation (9). We describe below how the Hellinger distance, for instance, is realized not only by the exponential loss (as described earlier), but also by many other surrogate loss functions. Additional examples can be found in [7].

**Illustrative examples.** Consider Hellinger distance, which is an $f$-divergence[2] with $f(u) = -2\sqrt{u}$. Augment the domain of $f$ with $f(u) = +\infty$ for $u < 0$. Following the prescription of Theorem 4(a), we first recover $\Psi$ from $f$:

$$\Psi(\beta) = f^*(-\beta) = \sup_{u \in \mathbb{R}}(-\beta u - f(u)) = \begin{cases} 1/\beta & \text{when } \beta > 0 \\ +\infty & \text{otherwise.} \end{cases}$$

Clearly, $u^* = 1$. Now if we choose $g(u) = e^{u-1}$, then we obtain the exponential loss $\phi(\alpha) = \exp(-\alpha)$. However, making the alternative choice $g(u) = u$, we obtain the function $\phi(\alpha) = 1/(\alpha+1)$ and $\phi(-\alpha) = \alpha+1$, which also realizes the Hellinger distance.

Recall that we have shown previously that the 0-1 loss induces the variational distance, which can be expressed as an $f$-divergence with $f_{\mathrm{var}}(u) = -2\min(u, 1)$ for $u \geq 0$. It is thus of particular interest to determine other loss functions that also lead to variational distance. If we augment the function $f_{\mathrm{var}}$ by defining $f_{\mathrm{var}}(u) = +\infty$ for $u < 0$, then we can recover $\Psi$ from $f_{\mathrm{var}}$ as follows:

$$\Psi(\beta) = f_{\mathrm{var}}^*(-\beta) = \sup_{u \in \mathbb{R}}(-\beta u - f_{\mathrm{var}}(u)) = \begin{cases} (2 - \beta)_+ & \text{when } \beta \geq 0 \\ +\infty & \text{when } \beta < 0. \end{cases}$$

Clearly $u^* = 1$. Choosing $g(u) = u$ leads to the hinge loss $\phi(\alpha) = (1 - \alpha)_+$, which is consistent with our earlier findings. Making the alternative choice $g(u) = e^{u-1}$ leads to a rather different loss—namely, $\phi(\alpha) = (2 - e^\alpha)_+$ for $\alpha \geq 0$ and $\phi(\alpha) = e^{-\alpha}$ for $\alpha < 0$—that also realizes the variational distance.

Using Theorem 4 it can be shown that an $f$-divergence is realizable by a margin-based surrogate loss if and only if it is symmetric [7]. Hence, the list of non-realizable $f$-divergences includes the $KL$ divergence $KL(\mu||\pi)$ (as well as $KL(\pi||\mu)$). The *symmetric* KL divergence $KL(\mu||\pi) + KL(\pi||\mu)$ is a realizable $f$-divergence. Theorem 4 allows us to construct all $\phi$ losses that realize it. One of them turns out to have the simple closed-form $\phi(\alpha) = e^{-\alpha} - \alpha$, but obtaining it requires some non-trivial calculations [7].

## 4 On comparison of loss functions and quantization schemes

The previous section was devoted to study of the correspondence between $f$-divergences and the optimal $\phi$-risk $R_\phi(Q)$ for a fixed experiment $Q$. Our ultimate goal, however, is that of choosing an optimal $Q$, a problem known as experimental design in the statistics literature [3]. One concrete application is the design of quantizers for performing decentralized detection [12, 6] in a sensor network.

In this section, we address the experiment design problem via the joint optimization of $\phi$-risk (or more precisely, its empirical version) over both the decision $\gamma$ and the choice of experiment $Q$ (hereafter referred to as a quantizer). This procedure raises the natural theoretical question: for what loss functions $\phi$ does such joint optimization lead to minimum Bayes risk? Note that the minimum here is taken over both the decision rule $\gamma$ and the space of experiments $Q$, so that this question is not covered by standard consistency results [13, 10, 2]. Here we describe how the results of the previous section can be leveraged to resolve this issue of consistency.

### 4.1 Universal equivalence

The connection between $f$-divergences and 0-1 loss can be traced back to seminal work on the comparison of experiments [3]. Formally, we say that the quantization scheme $Q_1$ *dominates* than $Q_2$ if $R_{bayes}(Q_1) \leq R_{bayes}(Q_2)$ for any prior probabilities $q \in (0, 1)$. We have the following theorem [3] (see also [7] for a short proof):

**Theorem 5.** $Q_1$ dominates $Q_2$ iff $I_f(\mu^{Q_1}, \pi^{Q_1}) \geq I_f(\mu^{Q_2}, \pi^{Q_2})$, for all convex functions $f$. The superscripts denote the dependence of $\mu$ and $\pi$ on the quantizer rules $Q_1, Q_2$.

Using Lemma 2, we can establish the following:

**Corollary 6.** $Q_1$ dominates $Q_2$ iff $R_\phi(Q_1) \leq R_\phi(Q_2)$ for any surrogate loss $\phi$.

One implication of Corollary 6 is that if $R_\phi(Q_1) \leq R_\phi(Q_2)$ for some loss function $\phi$, then $R_{bayes}(Q_1) \leq R_{bayes}(Q_2)$ for some set of prior probabilities on the labels $Y$. This fact justifies the use of a surrogate $\phi$-loss as a proxy for the 0-1 loss, at least for a certain subset of prior probabilities. Typically, however, the goal is to select the optimal experiment $Q$ for a pre-specified set of priors, in which context this implication is of limited use. We are thus motivated to consider a different method of determining which loss functions (or equivalently, $f$-divergences) lead to the same optimal experimental design as the 0-1 loss (respectively the variational distance). More generally, we are interested in comparing two arbitrary loss function $\phi_1$ and $\phi_2$, with corresponding divergences induced by $f_1$ and $f_2$ respectively:

**Definition 7.** *The surrogate loss functions $\phi_1$ and $\phi_2$ are* universally equivalent*, denoted by $\phi_1 \overset{u}{\approx} \phi_2$ (and $f_1 \overset{u}{\approx} f_2$), if for any $P(X, Y)$ and quantization rules $Q_1, Q_2$, there holds:*

$$R_{\phi_1}(Q_1) \leq R_{\phi_1}(Q_2) \Leftrightarrow R_{\phi_2}(Q_1) \leq R_{\phi_2}(Q_2). \tag{10}$$

The following result provides necessary and sufficient conditions for universal equivalence:

**Theorem 8.** *Suppose that $f_1$ and $f_2$ are differentiable a.e., convex functions that map $[0, +\infty)$ to $\mathbb{R}$. Then $f_1 \overset{u}{\approx} f_2$ if and only if $f_1(u) = cf_2(u) + au + b$ for some constants $a, b \in \mathbb{R}$ and $c > 0$.*

If we restrict our attention to convex and differentiable a.e. functions $f$, then it follows that all $f$-divergences univerally equivalent to the variational distance must have the form

$$f(u) = -c\min(u, 1) + au + b \qquad \text{with } c > 0. \tag{11}$$

As a consequence, the only $\phi$-loss functions universally equivalent to 0-1 loss are those that induce an $f$-divergence of this form (11). One well-known example of such a function is the hinge loss; more generally, Theorem 4 allows us to construct all such $\phi$.

## 4.2   Consistency in experimental design

The notion of universal equivalence might appear quite restrictive because condition (10) must hold for *any* underlying probability measure $P(X, Y)$. However, this is precisely what we need when $P(X, Y)$ is unknown. Assume that the knowledge about $P(X, Y)$ comes from an empirical data sample $(x_i, y_i)_{i=1}^n$.

Consider any algorithm (such as that proposed by Nguyen et al. [6]) that involves choosing a classifier-quantizer pair $(\gamma, Q) \in \Gamma \times \mathcal{Q}$ by minimizing an empirical version of $\phi$-risk:

$$\hat{R}_\phi(\gamma, Q) := \frac{1}{n} \sum_{i=1}^n \sum_z \phi(y_i \gamma(z)) Q(z|x_i).$$

More formally, suppose that $(\mathcal{C}_n, \mathcal{D}_n)$ is a sequence of increasing compact function classes such that $\mathcal{C}_1 \subseteq \mathcal{C}_2 \subseteq \ldots \subseteq \Gamma$ and $\mathcal{D}_1 \subseteq \mathcal{D}_2 \subseteq \ldots \subseteq \mathcal{Q}$. Let $(\gamma_n^*, Q_n^*)$ be an optimal solution to the minimization problem $\min_{(\gamma, Q) \in (\mathcal{C}_n, \mathcal{D}_n)} \hat{R}_\phi(\gamma, Q)$, and let $R_{bayes}^*$ denote the minimum Bayes risk achieved over the space of decision rules $(\gamma, Q) \in (\Gamma, \mathcal{Q})$. We call $R_{bayes}(\gamma_n^*, Q_n^*) - R_{bayes}^*$ the *Bayes error* of our estimation procedure. We say that such a procedure is *universally consistent* if the Bayes error tends to 0 as $n \to \infty$, i.e., for any (unknown) Borel probability measure $P$ on $X \times Y$,

$$\lim_{n \to \infty} R_{bayes}(\gamma_n^*, Q_n^*) - R_{bayes}^* = 0 \quad \text{in probability.}$$

When the surrogate loss $\phi$ is universally equivalent to 0-1 loss, we can prove that suitable learning procedures are indeed universally consistent. Our approach is based on the framework developed by various authors [13, 10, 2] for the case of ordinary classification, and using the strategy of decomposing the Bayes error into a combination of (a) *approximation error* introduced by the bias of the function classes $\mathcal{C}_n \subseteq \Gamma$: $\mathcal{E}_0(\mathcal{C}_n, \mathcal{D}_n) = \inf_{(\gamma, Q) \in (\mathcal{C}_n, \mathcal{D}_n)} R_\phi(\gamma, Q) - R_\phi^*$, where $R_\phi^* := \inf_{(\gamma, Q) \in (\Gamma, \mathcal{Q})} R_\phi(\gamma, Q)$; and (b) *estimation error* introduced by the variance of using finite sample size $n$, $\mathcal{E}_1(\mathcal{C}_n, \mathcal{D}_n) = \mathbb{E} \sup_{(\gamma, Q) \in (\mathcal{C}_n, \mathcal{D}_n)} |\hat{R}_\phi(\gamma, Q) - R_\phi(\gamma, Q)|$, where the expectation is taken with respect to the (unknown) probability measure $P(X, Y)$.

**Assumptions.**   Assume that the loss function $\phi$ is universally equivalent to the 0-1 loss. From Theorem 8, the corresponding $f$-divergence must be of the form $f(u) = -c\min(u, 1) + au + b$, for $a, b \in \mathbb{R}$ and $c > 0$. Finally, we also assume that $(a - b)(p - q) \geq 0$ and $\phi(0) \geq 0$.[3] In addition, for each $n = 1, 2, \ldots$, suppose that $M_n := \sup_{y,z} \sup_{(\gamma, Q) \in (\mathcal{C}_n, \mathcal{D}_n)} |\phi(y\gamma(z))| < +\infty$.

The following lemma plays a key role in our proof: it links the excess $\phi$-risk to the Bayes error when performing joint minimization:

**Lemma 9.** *For any $(\gamma, Q)$, we have $\frac{c}{2}(R_{bayes}(\gamma, Q) - R^*_{bayes}) \leq R_\phi(\gamma, Q) - R^*_\phi$.*

Finally, we can relate the Bayes error to the approximation error and estimation error, and provide general conditions for universal consistency:

**Theorem 10.** *(a) For any Borel probability measure $P$, with probability at least $1 - \delta$, there holds: $R_{bayes}(\gamma^*_n, Q^*_n) - R^*_{bayes} \leq \frac{2}{c}(2\mathcal{E}_1(\mathcal{C}_n, \mathcal{D}_n) + \mathcal{E}_0(\mathcal{C}_n, \mathcal{D}_n) + 2M_n\sqrt{2\ln(2/\delta)/n})$. (b) (Universal Consistency) If $\cup^\infty_{n=1}\mathcal{D}_n$ is dense in $\mathcal{Q}$ and if $\cup^\infty_{n=1}\mathcal{C}_n$ is dense in $\Gamma$ so that $\lim_{n\to\infty}\mathcal{E}_0(\mathcal{C}_n, \mathcal{D}_n) = 0$, and if the sequence of function classes $(\mathcal{C}_n, \mathcal{D}_n)$ grows sufficiently slowly enough so that $\lim_{n\to\infty}\mathcal{E}_1(\mathcal{C}_n, \mathcal{D}_n) = \lim_{n\to\infty}M_n\sqrt{\ln n/n} = 0$, there holds $\lim_{n\to\infty}R_{bayes}(\gamma^*_n, Q^*_n) - R^*_{bayes} = 0$ in probability.*

## 5 Conclusions

We have presented a general theoretical connection between surrogate loss functions and $f$-divergences. As illustrated by our application to decentralized detection, this connection can provide new domains of application for statistical learning theory. We also expect that this connection will provide new applications for $f$-divergences within learning theory; note in particular that bounds among $f$-divergences (of which many are known; see, e.g., [11]) induce corresponding bounds among loss functions.

## Footnotes

[1]Proofs are omitted from this manuscript for lack of space; see the long version of the paper [7] for proofs of all of our results.

[2]We consider $f$-divergences for two convex functions $f_1$ and $f_2$ to be equivalent if $f_1$ and $f_2$ are related by a linear term, i.e., $f_1 = cf_2 + au + b$ for some constants $c > 0, a, b$, because then $I_{f_1}$ and $I_{f_2}$ are different by a constant.

[3]These technical conditions are needed so that the approximation error due to varying $Q$ dominates the approximation error due to varying $\gamma$. Setting $a = b$ is sufficient.

## References

[1] S. M. Ali and S. D. Silvey. A general class of coefficients of divergence of one distribution from another. *J. Royal Stat. Soc. Series B*, 28:131–142, 1966.

[2] P. Bartlett, M. I. Jordan, and J. D. McAuliffe. Convexity, classification and risk bounds. *Journal of the American Statistical Association*, 2005. To appear.

[3] D. Blackwell. Equivalent comparisons of experiments. *Annals of Statistics*, 24(2):265–272, 1953.

[4] I. Csiszár. Information-type measures of difference of probability distributions and indirect observation. *Studia Sci. Math. Hungar*, 2:299–318, 1967.

[5] T. Kailath. The divergence and Bhattacharyya distance measures in signal selection. *IEEE Trans. on Communication Technology*, 15(1):52–60, 1967.

[6] X. Nguyen, M. J. Wainwright, and M. I. Jordan. Nonparametric decentralized detection using kernel methods. *IEEE Transactions on Signal Processing*, 53(11):4053–4066, 2005.

[7] X. Nguyen, M. J. Wainwright, and M. I. Jordan. On divergences, surrogate loss functions and decentralized detection. Technical Report 695, Department of Statistics, University of California at Berkeley, September 2005.

[8] H. V. Poor and J. B. Thomas. Applications of Ali-Silvey distance measures in the design of generalized quantizers for binary decision systems. *IEEE Trans. on Communications*, 25:893–900, 1977.

[9] G. Rockafellar. *Convex Analysis*. Princeton University Press, Princeton, 1970.

[10] I. Steinwart. Consistency of support vector machines and other regularized kernel machines. *IEEE Trans. Info. Theory*, 51:128–142, 2005.

[11] F. Topsoe. Some inequalities for information divergence and related measures of discrimination. *IEEE Transactions on Information Theory*, 46:1602–1609, 2000.

[12] J. Tsitsiklis. Extremal properties of likelihood-ratio quantizers. *IEEE Trans. on Communication*, 41(4):550–558, 1993.

[13] T. Zhang. Statistical behavior and consistency of classification methods based on convex risk minimization. *Annal of Statistics*, 53:56–134, 2004.
